# Posterior vs. Parameter Sparsity in Latent Variable Models

**João V. Graça**
L²F INESC-ID
Lisboa, Portugal

**Kuzman Ganchev**  **Ben Taskar**
University of Pennsylvania
Philadelphia, PA, USA

**Fernando Pereira**
Google Research
Mountain View, CA, USA

## Abstract

We address the problem of learning structured unsupervised models with moment sparsity typical in many natural language induction tasks. For example, in unsupervised part-of-speech (POS) induction using hidden Markov models, we introduce a bias for words to be labeled by a small number of tags. In order to express this bias of posterior sparsity as opposed to parametric sparsity, we extend the posterior regularization framework [7]. We evaluate our methods on three languages — English, Bulgarian and Portuguese — showing consistent and significant accuracy improvement over EM-trained HMMs, and HMMs with sparsity-inducing Dirichlet priors trained by variational EM. We increase accuracy with respect to EM by 2.3%-6.5% in a purely unsupervised setting as well as in a weakly-supervised setting where the closed-class words are provided. Finally, we show improvements using our method when using the induced clusters as features of a discriminative model in a semi-supervised setting.

## 1 Introduction

Latent variable generative models are widely used in inducing meaningful representations from unlabeled data. Maximum likelihood estimation is a standard method for fitting such models, but in most cases we are not so interested in the likelihood of the data as in the distribution of the latent variables, which we hope will capture regularities of interest without direct supervision. In this paper we explore the problem of biasing such unsupervised models to favor a novel kind of sparsity that expresses our expectations about the role of the latent variables. Many important language processing tasks (tagging, parsing, named-entity classification) involve classifying events into a large number of possible classes, where each event type can have just a few classes. We extend the posterior regularization framework [7] to achieve that kind of *posterior* sparsity on the unlabeled training data. In unsupervised part-of-speech (POS) tagging, a well studied yet challenging problem, the new method consistently and significantly improves performance over a non-sparse baseline and over a variational Bayes baseline with a Dirichlet prior used to encourage sparsity [9, 4].

A common approach to unsupervised POS tagging is to train a hidden Markov model where the hidden states are the possible tags and the observations are word sequences. The model is typically trained with the expectation-maximization (EM) algorithm to maximize the likelihood of the observed sentences. Unfortunately, while supervised training of HMMs achieves relatively high accuracy, the unsupervised models tend to perform poorly. One well-known reason for this is that EM tends to allow each word to be generated by most hidden states some of the time. In reality, we would like most words to have a small number of possible tags. To solve this problem, several studies [14, 17, 6] investigated weakly-supervised approaches where the model is given the list of possible tags for each word. The task is then to disambiguate among the possible tags for each word type. Recent work has made use of smaller dictionaries, trying to model the set of possible tags for each word [18, 5], or use a small number of "prototypes" for each tag [8]. All these approaches initialize the model in a way that encourages sparsity by zeroing out impossible tags. Although this

has worked extremely well for the weakly-supervised case, we are interested in the setting where we have only high-level information about the model: we know that the distribution over the latent variables (such as POS tags) should be sparse. This has been explored in a Bayesian setting, where a prior is used to encourage sparsity in the model *parameters* [4, 9, 6]. This sparse prior, which prefers each tag to have few word types associated with it, indirectly achieves sparsity over the *posteriors*, meaning each word type should have few possible tags. Our method differs in that it encourages sparsity in the model posteriors, more directly encoding the desiderata. Additionally our method can be applied to log-linear models where sparsity in the parameters leads to dense posteriors. Sparsity at this level has already been suggested before under a very different model[18].

We use a first-order HMM as our model to compare the different training conditions: classical expectation-maximization (EM) training without modifications to encourage sparsity, the sparse prior used by [9] with variational Bayes EM (VEM), and our sparse posterior regularization (Sparse). We evaluate these methods on three languages, English, Bulgarian and Portuguese. We find that our method consistently improves performance with respect to both baselines in a completely unsupervised scenario, as well as in a weakly-supervised scenario where the tags of closed-class words are supplied. Interestingly, while VEM achieves a state size distribution (number of words assigned to hidden states) that is closer to the empirical tag distribution than EM and Sparse its state-token distribution is a worse match to the empirical tag-token distribution than the competing methods. Finally, we show that states assigned by the model are useful as features for a supervised POS tagger.

## 2    Posterior Regularization

In order to express the desired preference for posterior sparsity, we use the posterior regularization (PR) framework [7], which incorporates side information into parameter estimation in the form of linear constraints on posterior expectations. This allows tractable learning and inference even when the constraints would be intractable to encode directly in the model, for instance to enforce that each hidden state in an HMM is used only once in expectation. Moreover, PR can represent prior knowledge that cannot be easily expressed as priors over model parameters, like the constraint used in this paper. PR can be seen as a penalty on the standard marginal likelihood objective, which we define first:

$$\textbf{Marginal Likelihood:} \quad \mathcal{L}(\theta) = \widehat{\textbf{E}}[-\log p_\theta(\textbf{x})] = \widehat{\textbf{E}}[-\log \sum_{\textbf{z}} p_\theta(\textbf{z}, \textbf{x})]$$

over the parameters $\theta$, where $\widehat{\textbf{E}}$ is the empirical expectation over the unlabeled sample $\textbf{x}$, and $\textbf{z}$ are the hidden states. This standard objective may be regularized with a parameter prior $-\log p(\theta) = C(\theta)$, for example a Dirichlet.

Posterior information in PR is specified with sets $\mathcal{Q}_\textbf{x}$ of distributions over the hidden variables $\textbf{z}$ defined by linear constraints on feature expectations:

$$\mathcal{Q}_\textbf{x} = \{q(\textbf{z} \mid \textbf{x}) : \textbf{E}_q[\textbf{f}(\textbf{x}, \textbf{z})] \leq \textbf{b}\}. \tag{1}$$

The marginal log-likelihood of a model is then penalized with the KL-divergence between the desired distributions $\mathcal{Q}_\textbf{x}$ and the model, $\text{KL}(\mathcal{Q}_\textbf{x} \parallel p_\theta(\textbf{z}|\textbf{x})) = \min_{q \in \mathcal{Q}_\textbf{x}} \text{KL}(q(\textbf{z}) \parallel p_\theta(\textbf{z}|\textbf{x}))$. The revised learning objective minimizes:

$$\textbf{PR Objective:} \quad \mathcal{L}(\theta) + C(\theta) + \widehat{\textbf{E}}[\text{KL}(\mathcal{Q}_\textbf{x} \parallel p_\theta(\textbf{z}|\textbf{x}))]. \tag{2}$$

Since the objective above is not convex in $\theta$, PR estimation relies on an EM-like lower-bounding scheme for model fitting, where the E step computes a distribution $q(\textbf{z}|\textbf{x})$ over the latent variables and the M step minimizes negative marginal likelihood under $q(\textbf{z}|\textbf{x})$ plus parameter regularization:

$$\text{M-Step:} \quad \min_\theta \ \widehat{\textbf{E}}\left[\textbf{E}_q[-\log p_\theta(\textbf{x}, \textbf{z})]\right] + C(\theta) \tag{3}$$

In a standard E step, $q$ is the posterior over the model hidden variables given current $\theta$: $q(\textbf{z}|\textbf{x}) = p_\theta(\textbf{z}|\textbf{x})$. However, in PR, $q$ is a projection of the posteriors onto the constraint set $\mathcal{Q}_\textbf{x}$ for each example $\textbf{x}$:

$$\underset{q}{\arg\min} \text{KL}(q(\textbf{z}|\textbf{x}) \parallel p_\theta(\textbf{z}|\textbf{x})) \quad \text{s.t. } \textbf{E}_q[\textbf{f}(\textbf{x}, \textbf{z})] \leq \textbf{b}. \tag{4}$$

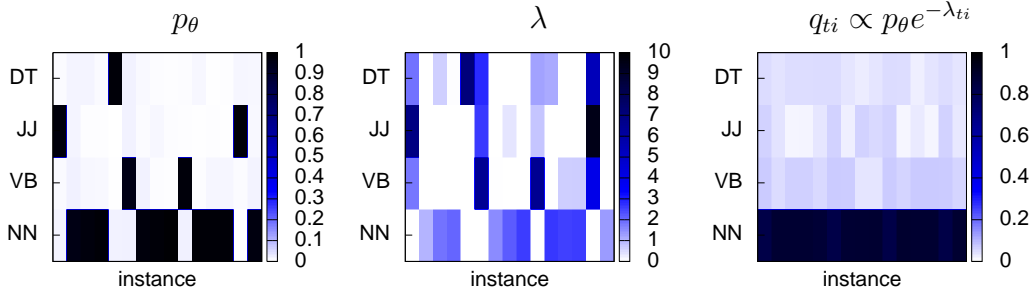

Figure 1: An illustration of $\ell_1/\ell_\infty$ regularization. Left panel: initial tag distributions (columns) for 15 instances of a word. Middle panel: optimal regularization parameters $\lambda$, each row sums to $\sigma = 20$. Right panel: $q$ concentrates the posteriors for all instances on the NN tag, reducing the $\ell_1/\ell_\infty$ norm from just under 4 to a little over 1.

The new posteriors $q(\mathbf{z}|\mathbf{x})$ are used to compute sufficient statistics for this instance and hence to update the model's parameters in the M step. The optimization problem in Equation 4 can be solved efficiently in dual form:

$$\arg\min_{\lambda \geq 0} \mathbf{b}^\top \lambda + \log \sum_{\mathbf{z}} p_\theta(\mathbf{z}|\mathbf{x}) \exp\left\{-\lambda^\top \mathbf{f}(\mathbf{x}, \mathbf{z})\right\}. \tag{5}$$

Given $\lambda$, the primal solution is $q(\mathbf{z}|\mathbf{x}) = p_\theta(\mathbf{z}|\mathbf{x}) \exp\{-\lambda^\top \mathbf{f}(\mathbf{x}, \mathbf{z})\}/Z$, where $Z$ is a normalization constant. There is one dual variable per expectation constraint, which can be optimized by projected gradient descent where gradient for $\lambda$ is $\mathbf{b} - \mathbf{E}_q[\mathbf{f}(\mathbf{x}, \mathbf{z})]$. Gradient computation involves an expectation under $q(\mathbf{z}|\mathbf{x})$ that can be computed efficiently if the features $\mathbf{f}(\mathbf{x}, \mathbf{z})$ factor in the same way as the model $p_\theta(\mathbf{z}|\mathbf{x})$ [7].

## 3  Relaxing Posterior Regularization

In this work, we modify PR so that instead of hard constraints on $q(\mathbf{z} \mid \mathbf{x})$, it allows the constraints to be relaxed at a cost specified by a penalty. This relaxation can allow combining multiple constraints without having to explicitly ensure that the constraint set remains non-empty. Additionally, it will be useful in dealing with the $\ell_1/\ell_\infty$ constraints we need. If those were incorporated as hard constraints, the dual objective would become non-differentiable, making the optimization (somewhat) more complicated. Using soft constraints, the non-differentiable portion of the dual objective turns into simplex constraints on the dual variables, allowing us to use an efficient projected gradient method. For soft constraints, Equation 4 is replaced by

$$\arg\min_{q,b} \mathrm{KL}(q \parallel p) + R(\mathbf{b}) \quad \text{s.t.} \quad \mathbf{E}_q[\mathbf{f}(\mathbf{x}, \mathbf{z})] \leq \mathbf{b} \tag{6}$$

where $\mathbf{b}$ is the constraint vector, and $R(\mathbf{b})$ penalizes overly lax constraints. For POS tagging, we will design $R(\mathbf{b})$ to encourage each word type to be observed with a small number of POS tags in the projected posteriors $q$. The overall objective minimized can be shown to be:

**Soft PR Objective:** $\arg\min_{\theta,q,\mathbf{b}} \mathcal{L}(\theta) + C(\theta) + \widehat{\mathbf{E}}[\mathrm{KL}(q \parallel p_\theta) + R(\mathbf{b})] \quad \text{s.t.} \quad \mathbf{E}_q[\mathbf{f}(\mathbf{x}, \mathbf{z})] \leq \mathbf{b}.$
$$\tag{7}$$

### 3.1  $\ell_1/\ell_\infty$ regularization

We now choose the posterior constraint regularizer $R(\mathbf{b})$ to encourage each word to be associated with only a few parts of speech. Let feature $f_{wti}$ have value 1 whenever the $i^{th}$ occurrence of word $w$ has part of speech tag $t$. For every word $w$, we would like there to be only a few POS tags $t$ such that there are occurrences $i$ where $t$ has nonzero probability. This can be achieved if it "costs" a lot to allow an occurrence of a word to take a tag, but once that happens, it should be "free" for other occurrences of the word to receive that same tag. More precisely, we would like the sum ($\ell_1$ norm) over tags $t$ and word types $w$ of the maxima ($\ell_\infty$ norm) of the expectation of taking tag $t$

over all occurrences of $w$ to be small. Table 1 shows the value of the $\ell_1/\ell_\infty$ sparsity measure for three different corpora, comparing fully supervised HMM and fully unsupervised HMM learned with standard EM, with standard EM having 3-4 times larger value of $\ell_1/\ell_\infty$ than the supervised. This discrepancy is what our PR objective is attempting to eliminate.

Formally, the E-step of our approach is expressed by the objective:

$$\min_{q,c_{wt}} \; \mathrm{KL}(q \parallel p_\theta) + \sigma \sum_{wt} c_{wt} \quad \text{s.\,t.} \quad \mathbf{E}_q[f_{wti}] \leq c_{wt} \tag{8}$$

where $\sigma$ is the strength of the regularization. Note that setting $\sigma = 0$ we are back to normal EM where $q$ is the model posterior distribution. As $\sigma \to \infty$, the constraints force each occurrence of a word type to have the same posterior distribution, effectively reducing the mode to a 0th-order Markov chain in the E step.

The dual of this objective has a very simple form (see supplementary material for derivation):

$$\max_{\boldsymbol{\lambda} \geq 0} \; -\log\left(\sum_z p_\theta(\mathbf{z}) \exp(-\boldsymbol{\lambda} \cdot \mathbf{f}(\mathbf{z}))\right) \quad \text{s.\,t.} \quad \sum_i \lambda_{wti} \leq \sigma \tag{9}$$

where $\mathbf{z}$ ranges over assignments to the hidden tag variables for all of the occurrences in the training data, $\mathbf{f}(\mathbf{z})$ is the vector of $f_{wti}$ feature values for assignment $\mathbf{z}$, $\boldsymbol{\lambda}$ is the vector of dual parameters $\lambda_{wti}$, and the primal parameters are $q(\mathbf{z}) \propto p_\theta(\mathbf{z}) \exp\left(-\boldsymbol{\lambda} \cdot \mathbf{f}(\mathbf{z})\right)$. This can be computed by projected gradient, as described by Bertsekas [3].

Figure 1 illustrates how the $\ell_1/\ell_\infty$ norm operates on a toy example. For simplicity suppose we are only regularizing one word and our model $p_\theta$ is just a product distribution over 15 instances of the word. The left panel in Figure 1 shows the posteriors under $p_\theta$. We would like to concentrate the posteriors on a small subset of rows. The center panel of the figure shows the $\boldsymbol{\lambda}$ values determined by Equation 9, and the right panel shows the projected distribution $q$, which concentrates most of the posterior on the bottom row. Note that we are not requiring the posteriors to be sparse, which would be equivalent to preferring that the distribution is peaked; rather, we want a word to concentrate its tag posterior on a few tags across all instances of the word. Indeed, most of the instances (columns) become less peaked than in the original posterior to allow posterior mass to be redistributed away from the outlier tags. Since they are more numerous than the outliers, they moved less. This also justifies only regularizing relatively frequent events in our model.

## 4 Bayesian Estimators

Recent advances in inference methods for sparsifying Bayesian estimation have been applied to unsupervised POS tagging [4, 9, 6]. In the Bayesian setting, preference for sparsity is expressed as a prior distribution over model structures and parameters, rather than as constraints on feature posteriors. To compare these two approaches, in Section 5 we compare our method to a Bayesian approach proposed by Johnson [9], which relies on a Dirichlet prior to encourage sparsity in a first-order HMM for POS tagging. The complete description of the model is:

$$\begin{aligned}
\theta_i &\sim Dir(\alpha_i) & \phi_i &\sim Dir(\lambda_i) \\
P(t_i | t_{t-1} = tag) &\sim Multi(\theta_i) & P(w_i | t_i = tag) &\sim Multi(\phi_i)
\end{aligned}$$

Here, $\alpha_i$ controls sparsity over the state transition matrix and $\lambda_i$ controls the sparsity of state emission probabilities. Johnson [9] notes that $\alpha_i$ does not influence the model that much. In contrast, as $\lambda_i$ approaches zero, it encourages the model to have highly skewed $P(w_i | t_i = tag)$ distributions, that is, each tag is encouraged to generate a few words with high probability, and the rest with very low probability. This is not exactly the constraint we would like to enforce: there are some POS tags that generate many different words with relatively high probability (for example, nouns and verbs), while each word is associated with a small number of tags. This difference is one possible explanation for the relatively worse performance of this prior compared to our method.

Johnson [9] describes two approaches to learn the model parameters: a component-wise Gibbs sampling scheme (GS) and a variational Bayes (VB) approximation using a mean field. Since Johnson [9] found VB worked much better than GS, we use VB in our experiments. Additionally, VB is particularly simple to implement, consisting only a small modification to the M-Step of the EM algorithm. The Dirichlet prior hyper-parameters are added to the expected counts and passed through

a squashing function (exponential of the Digamma function) before being normalized. We refer the reader to the original paper for more detail (see also `http://www.cog.brown.edu/~mj/Publications.htm` for a bug fix in the Digamma function implementation).

## 5   Experiments

We now compare first-order HMMs trained using the three methods described earlier: the classical EM algorithm (EM), our $\ell_1/\ell_\infty$ posterior regularization based method (Sparse), and the model presented in Section 4 (VEM). Models were trained and tested on all available data of three corpora: the Wall Street Journal portion of the Penn treebank [13] using the reduced tag set of 17 tags [17] (PTB17); the Bosque subset of the Portuguese Floresta Sinta(c)tica Treebank [1] used for the ConLL X shared task on dependency parsing (PT-CoNLL); and the Bulgarian BulTreeBank [16] (BulTree) with the 12 coarse tags. We also report results on the full Penn treebank tag set in the supplementary materials. All words that occurred only once were replaced by the token "unk". To measure model sparsity, we compute the average $\ell_1/\ell_\infty$ norm over words occurring more than 10 times (denoted 'L1LMax' in our figures). Table 1 gives statistics for each corpus as well as the sparsity for a first-order HMM trained using the labeled data and using standard EM with unlabeled data.

|           | Types | Tokens | Unk  | Tags | Sup. $\ell_1/\ell_\infty$ | EM $\ell_1/\ell_\infty$ |
|-----------|-------|--------|------|------|---------------------------|-------------------------|
| PT-Conll  | 11293 | 206678 | 8.5% | 22   | 1.14                      | 4.57                    |
| BulTree   | 12177 | 174160 | 10%  | 12   | 1.04                      | 3.51                    |
| PTB17     | 23768 | 950028 | 2%   | 17   | 1.23                      | 3.97                    |

Table 1: Corpus statistics. All words with only one occurrence where replaced by the 'unk' token. The third column shows the percentage of tokens replaced. Sup. $\ell_1/\ell_\infty$ is the value of the sparsity measure for a fully supervised HMM trained on all available data and EM $\ell_1/\ell_\infty$ is the value of the sparsity measure for a fully unsupervised HMM trained using standard EM on all available data.

Following Gao and Johnson [4], the parameters were initialized with a "pseudo E step" as follows: we filled the expected count matrices with numbers $1 + X \times U(0,1)$, where $U(0,1)$ is a random number between 0 and 1 and $X$ is a parameter. These matrices are then fed to the M step; the resulting "random" transition and emission probabilities are used for the first real E step. For VEM, $X$ was set to 0.0001 (almost uniform) since this showed a significant improvement in performance. On the other hand, EM showed less sensitivity to initialization, and we used $X = 1$ which resulted in the best results. The models were trained for 200 iterations as longer runs did not significantly change the results (models converge before 100 iterations). For VEM we tested 4 different prior combinations, (all combinations of $10^{-1}$ and $10^{-3}$ for emission prior and transition prior), based on Johnson's results [9]. As previously noted, changing the transition priors does not affect the

| Estimator | PT-Conll | | BG | | PTB17 | |
|-----------|----------|-----|-----|-----|-------|-----|
|           | 1-Many   | 1-1 | 1-Many | 1-1 | 1-Many | 1-1 |
| EM              | 64.0(1.2)  | 40.4(3.0) | 59.4(2.2)  | 42.0(3.0) | 67.5(1.3)   | 46.4(2.6)   |
| VEM($10^{-1}$)  | 60.4(0.6)  | **51.1(2.3)** | 54.9(3.1)  | 46.4(3.0) | 68.2(0.8)*  | **52.8(3.5)** |
| VEM($10^{-4}$)  | 63.2(1.0)* | 48.1(2.2) | 56.1(2.8)  | 43.3(1.7)* | 67.3(0.8)*  | 49.6(4.3)   |
| Sparse (10)     | 68.5(1.3)  | 43.3(2.2) | 65.1(1.0)  | 48.0(3.3) | 69.5(1.6)   | 50.0(3.5)   |
| Sparse (32)     | **69.2(0.9)** | 43.2(2.9) | **66.0(1.8)** | 48.7(2.2) | **70.2(2.2)** | 49.5(2.0) |
| Sparse (100)    | 68.3(2.1)  | 44.5(2.4) | 65.9(1.6)  | **48.9(2.8)** | 68.7(1.1)   | 47.8(1.5)*  |

Table 2: Average accuracy (standard deviation in parentheses) over 10 different runs (random seeds identical across models) for 200 iterations. 1-Many and 1-1 are the two hidden-state to POS mappings described in the text. All models are first order HMMs: EM trained using expectation maximization, VEM trained using variational EM observation priors shown in parentheses, Sparse trained using PR with the constraint strength ($\sigma$) in parentheses. **Bold** indicates the best value for each column. All results except those starred are significant (p=0.005) on a paired t-test against the EM model.

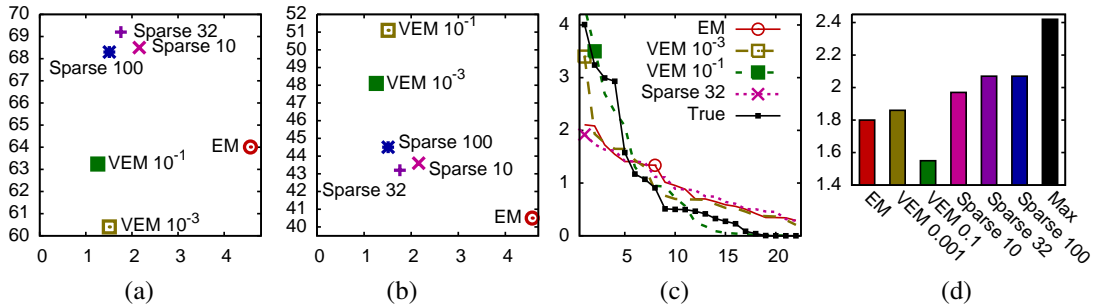

Figure 2: Detailed visualizations of the results on the PT-Conll corpus. (a) 1-many accuracy vs $\ell_1/\ell_\infty$, (b) 1-1 accuracy vs $\ell_1/\ell_\infty$, (c) tens of thousands of tokens assigned to hidden state vs rank, (d) mutual information in bits between gold tag distribution and hidden state distribution.

results, so we only report results for different emission priors. Later work [4] considered a wider range of values but did not identify definitely better choices. Sparse was initialized with the parameters obtained by running EM for 30 iterations, followed by 170 iterations of the new training procedure. Predictions were obtained using posterior decoding since this consistently showed small improvements over Viterbi decoding.

We evaluate the accuracy of the models using two established mappings between hidden states and POS tags: **1-Many** maps each hidden state to the tag with which it co-occurs the most; **1-1** [8] greedily picks a tag for each state under the constraint of never using the same tag twice. This results in an approximation of the optimal 1-1 mapping. If the numbers of hidden states and tags are not the same, some hidden states will be unassigned (and hence always wrong) or some tags not used. In all our experiments the number of hidden states is the same as the number of POS tags.

Table 2 shows the accuracy of the different methods averaged over 15 different random parameter initializations. Comparing the methods for each of the initialization points individually, our $\ell_1/\ell_\infty$ regularization always outperforms EM baseline model on both metrics, and always outperforms VEM using 1-Many mapping, while for the 1-1 mapping our method outperforms VEM roughly half the time. The improvements are consistent for different constraint strength values.

Figure 2 shows detailed visualizations of the behavior of the different methods on the PT-Conll corpus. The results for the other corpora are qualitatively similar, and are reported in the supplemental material. The left two plots show scatter graphs of accuracy with respect to $\ell_1/\ell_\infty$ value, where accuracy is measured with either the 1-many mapping (left) or 1-1 mapping (center). We see that Sparse is much better using the 1-many mapping and worse using the 1-1 mapping than VEM, even though they achieve similar $\ell_1/\ell_\infty$. The third plot shows the number of tokens assigned to each hidden state at decoding time, in frequency rank order. While both EM and Sparse exhibit a fast decrease in the size of the states, VEM more closely matches the power law-like distribution achieved by the gold labels. This difference explains the improvement on the 1-1 mapping, where VEM is assigning larger size states to the most frequent tags. However, VEM achieves this power law distribution at the expense of the mutual information with the gold labels as we see in the rightmost plot. From all methods, VEM has the lowest mutual information, while Sparse has the highest.

## 5.1 Closed-class words

We now consider the case where some supervision has been given in the form of a list of the closed-class words for the language, along with POS tags. Example closed classes are punctuation, pronouns, possessive markers, while open classes would include nouns, verbs, and adjectives. (See the supplemental materials for details.) We assume that we are given the POS tags of closed classes along with the words in each closed class. In the models, we set the emission probability from a closed-class tag to any word not in its class to zero. Also, any word appearing in a closed class is assumed to have zero probability of being generated by an open-class tag. This improves performance significantly for all languages, but our sparse training procedure is still able to outperform EM training significantly as shown in Table 3. Note, for these experiments we do not use an unknown word, since doing so for closed-class words would allow closed class tags to generate unknown words.

| Estimator | PT-Conll | | BulTree | | PTB-17 | |
|---|---|---|---|---|---|---|
| | 1-Many | 1-1 | 1-Many | 1-1 | 1-Many | 1-1 |
| EM | 72.5(1.7) | 52.6(4.2) | 77.9(1.7) | 65.4(2.8) | 76.7(0.9) | 61.1(1.8) |
| Sparse (32) | **75.3(1.2)** | **57.5(5.0)** | **82.4(1.2)** | **69.5(1.3)** | **78.0(1.6)** | **62.2(2.0)** |

Table 3: Results with given closed-class tags, using posterior decoding, and projection at test time.

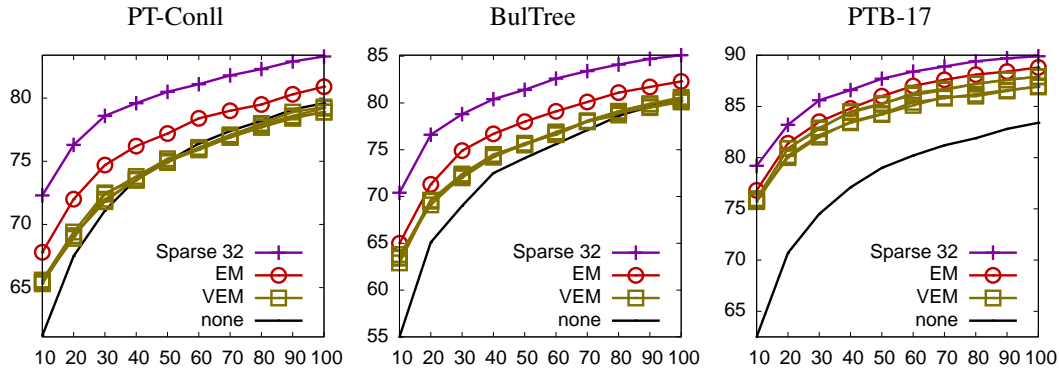

Figure 3: Accuracy of a supervised classifier when trained using the output of various unsupervised models as features. Vertical axis: accuracy, Horizontal axis: number of labeled sentences.

## 5.2 Supervised POS tagging

As a further comparison of the models trained using the different methods, we use them to generate features for a supervised POS tagger. The basic supervised model has features for the identity of the current token as well as suffixes of length 2 and 3. We augment these features with the state identity for the current token, based on the automatically generated models. We train the supervised model using averaged perceptron for 20 iterations.

For each unsupervised training procedure (EM, Sparse, VEM) we train 10 models using different random initializations and got 10 state identities per training method for each token. We then add these cluster identities as features to the supervised model. Figure 3 shows the average accuracy of the supervised model as we vary the type of unsupervised features. The average is taken over 10 random samples for the training set at each training set size. We can see from Figure 3 that using our method or EM always improves performance relative to the baseline features (labeled "none" in the figure). VEM always under performs EM and for larger amounts of training data, the VEM features appear not to be useful. This should not be surprising given that VEM has very low mutual information with the gold labeling.

## 6 Related Work

Our learning method is very closely related to the work of Mann and McCallum [11, 12], who concurrently developed the idea of using penalties based on posterior expectations of features to guide learning. They call their method generalized expectation (GE) constraints or alternatively expectation regularization. In the original GE framework, the posteriors of the model are regularized directly. For equality constraints, our objective would become:

$$\arg\max_{\theta} \quad \mathcal{L}(\theta) - \mathbf{E}_D[R(\mathbf{E}_\theta[f])]. \tag{10}$$

Notice that there is no intermediate distribution $q$. For some kinds of constraints this objective is difficult to optimize in $\theta$ and in order to improve efficiency Bellare *et al.* [2] propose interpreting the PR framework as an approximation to the GE objective in Equation 10. They compare the two frameworks on several datasets and find that performance is similar, and we suspect that this would be true for the sparsity constraints also. Liang *et al.* [10] cast the problem of incorporating partial information about latent variables into a Bayesian framework using "measurements," and they propose active learning for acquiring measurements to reduce uncertainty.

Recently, Ravi *et al.* [15] show promising results in weakly-supervised POS tagging, where a tag dictionary is provided. This method first searches, using integer programming, for the smallest grammar (in terms of unique transitions between tags) that explains the data. This sparse grammar and the dictionary are provided as input for training an unsupervised HMM. Results show that using a sparse grammar, hence enforcing sparsity over possible sparsity transitions leads to better results. This method is different from ours in the sense that our method focuses on learning the sparsity pattern they their method uses as input.

# 7 Conclusion

We presented a new regularization method for unsupervised training of probabilistic models that favors a kind of sparsity that is pervasive in natural language processing. In the case of part-of-speech induction, the preference can be summarized as "each word occurs as only a few different parts-of-speech," but the approach is more general and could be applied to other tasks. For example, in grammar induction, we could favor models where only a small number of production rules have non-zero probability for each child non-terminal.

Our method uses the posterior regularization framework to specify preferences about model posteriors directly, without having to say how these should be encoded in model parameters. This means that the sparse regularization penalty could be used for a log-linear model, where sparse parameters do not correspond to posterior sparsity.

We evaluated the new regularization method on the task of unsupervised POS tagging, encoding the prior knowledge that each word should have a small set of tags as a mixed-norm penalty. We compared our method to a previously proposed Bayesian method (VEM) for encouraging sparsity of model parameters [9] and found that ours performs better in practice. We explain this advantage by noting that VEM encodes a preference that each POS tag should generate a few words, which goes in the wrong direction. In reality, in POS tagging (as in several other language processing task), a few event types (tags) (such the *NN* for POS tagging) generate the bulk of the word occurrences, but each word is only associated with a few tags. Even when some supervision was provided with through closed class lists, our regularizer still improved performance over the other methods.

An analysis of sparsity shows that both VEM and Sparse achieve a similar posterior sparsity as measured by the $\ell_1/\ell_\infty$ metric. While VEM models better the empirical sizes of states (tags), the states it assigns have lower mutual information to the true tags, suggesting that parameter sparsity is not as good at generating good tag assignments. In contrast, Sparse's sparsity seems to help build a model that contains more information about the correct tag assignments.

Finally, we evaluated the worth of states assigned by unsupervised learning as features for supervised tagger training with small training sets. These features are shown to be useful in most conditions, especially those created by Sparse. The exceptions are some of the annotations provided by VEM which actually hinder the performance, confirming that its lower mutual information states are not so informative.

In future work, we would like to evaluate the usefulness of these sparser annotations for downstream tasks, for example determining whether Sparse POS tags are better for unsupervised parsing. Finally, we would like to apply the $\ell_1/\ell_\infty$ posterior regularizer to other applications such as unsupervised grammar induction where we would like sparsity in production rules. Similarly, it would be interesting to use this to regularize a log-linear model, where parameter sparsity does not achieve the same goal.

**Acknowledgments**

J. V. Graça was supported by a fellowship from Fundação para a Ciência e Tecnologia (SFRH/ BD/ 27528/ 2006). K. Ganchev was supported by ARO MURI SUBTLE W911NF-07-1-0216 The authors would like to thank Mark Johnson and Jianfeng Gao for their help in reproducing the VEM results.

# References

[1] S. Afonso, E. Bick, R. Haber, and D. Santos. Floresta Sinta(c)tica: a treebank for Portuguese. In *In Proc. LREC*, pages 1698–1703, 2002.

[2] K. Bellare, G. Druck, and A. McCallum. Alternating projections for learning with expectation constraints. In *In Proc. UAI*, 2009.

[3] D.P. Bertsekas, M.L. Homer, D.A. Logan, and S.D. Patek. Nonlinear programming. *Athena scientific*, 1995.

[4] Jianfeng Gao and Mark Johnson. A comparison of Bayesian estimators for unsupervised Hidden Markov Model POS taggers. In *In Proc. EMNLP*, pages 344–352, Honolulu, Hawaii, October 2008. ACL.

[5] Y. Goldberg, M. Adler, and M. Elhadad. Em can find pretty good hmm pos-taggers (when given a good start). *In Proc. ACL*, pages 746–754, 2008.

[6] S. Goldwater and T. Griffiths. A fully bayesian approach to unsupervised part-of-speech tagging. In *In Proc. ACL*, volume 45, page 744, 2007.

[7] J. Graça, K. Ganchev, and B. Taskar. Expectation maximization and posterior constraints. In *In Proc. NIPS*. MIT Press, 2008.

[8] A. Haghighi and D. Klein. Prototype-driven learning for sequence models. In *In Proc. NAACL*, pages 320–327, 2006.

[9] M Johnson. Why doesn't EM find good HMM POS-taggers. In *In Proc. EMNLP-CoNLL*, 2007.

[10] P. Liang, M. I. Jordan, and D. Klein. Learning from measurements in exponential families. In *In proc. ICML*, 2009.

[11] G. Mann and A. McCallum. Simple, robust, scalable semi-supervised learning via expectation regularization. In *Proc. ICML*, 2007.

[12] G. Mann and A. McCallum. Generalized expectation criteria for semi-supervised learning of conditional random fields. In *In Proc. ACL*, pages 870 – 878, 2008.

[13] M.P. Marcus, M.A. Marcinkiewicz, and B. Santorini. Building a large annotated corpus of English: The Penn Treebank. *Computational linguistics*, 19(2):313–330, 1993.

[14] B. Merialdo. Tagging English text with a probabilistic model. *Computational linguistics*, 20(2):155–171, 1994.

[15] Sujith Ravi and Kevin Knight. Minimized models for unsupervised part-of-speech tagging. In *In Proc. ACL*, 2009.

[16] Kiril Simov, Petya Osenova, Milena Slavcheva, Sia Kolkovska, Elisaveta Balabanova, Dimitar Doikoff, Krassimira Ivanova, Alexander Simov, Er Simov, and Milen Kouylekov. Building a linguistically interpreted corpus of bulgarian: the bultreebank. In *In Proc. LREC*, page pages, 2002.

[17] N.A. Smith and J. Eisner. Contrastive estimation: Training log-linear models on unlabeled data. In *In Proc. ACL*, pages 354–362, 2005.

[18] K. Toutanova and M. Johnson. A Bayesian LDA-based model for semi-supervised part-of-speech tagging. *In Proc. NIPS*, 20, 2007.

